# Information Bottleneck for Gaussian Variables

**Gal Chechik**$^*$   **Amir Globerson**$^*$   **Naftali Tishby**   **Yair Weiss**
{*ggal,gamir,tishby,yweiss*}*@cs.huji.ac.il*
School of Computer Science and Engineering and
The Interdisciplinary Center for Neural Computation
The Hebrew University of Jerusalem, 91904, Israel
$^*$ Both authors contributed equally

## Abstract

The problem of extracting the relevant aspects of data was addressed through the *information bottleneck* (IB) method, by (soft) clustering one variable while preserving information about another - *relevance* - variable. An interesting question addressed in the current work is the extension of these ideas to obtain continuous representations that preserve relevant information, rather than discrete clusters. We give a formal definition of the general continuous IB problem and obtain an analytic solution for the optimal representation for the important case of multivariate Gaussian variables. The obtained optimal representation is a noisy linear projection to eigenvectors of the normalized correlation matrix $\Sigma_{x|y}\Sigma_x^{-1}$, which is also the basis obtained in Canonical Correlation Analysis. However, in Gaussian IB, the compression tradeoff parameter uniquely determines the dimension, as well as the scale of each eigenvector. This introduces a novel interpretation where solutions of different ranks lie on a continuum parametrized by the compression level. Our analysis also provides an analytic expression for the optimal tradeoff - the information curve - in terms of the eigenvalue spectrum.

## 1  Introduction

Extracting relevant aspects of complex data is a fundamental task in machine learning and statistics. The problem is often that the data contains many structures, which make it difficult to define which of them are relevant and which are not in an unsupervised manner. For example, speech signals may be characterized by their volume level, pitch, or content; pictures can be ranked by their luminosity level, color saturation or importance with regard to some task.

This problem was principally addressed by the information bottleneck (IB) approach [1]. Given the joint distribution of a "source" variable $X$ and another "relevance" variable $Y$, IB operates to compress $X$, while preserving information about $Y$. The variable $Y$ thus implicitly defines what is relevant in $X$ and what isn't. Formally, this is cast as the following variational problem

$$\min_{p(t|x)} \mathcal{L} : \mathcal{L} \equiv I(X;T) - \beta I(T;Y) \tag{1}$$

where $T$ represents the compression of $X$ via the conditional distributions $p(t|x)$, while the information that $T$ maintains on $Y$ is captured by $p(y|t)$. The positive parameter $\beta$ determines the tradeoff between compression and preserved relevant information, as the Lagrange multiplier for the constrained optimization problem $\min_{p(t|x)} I(X;T) - \beta\left(I(T;Y) - const\right)$.

The information bottleneck approach has been applied so far mainly to categorical variables, with a discrete $T$ that represents (soft) clusters of $X$. It has been proved useful for a range of applications from documents clustering, to gene expression analysis (see [2] for review and references). However, its general information theoretic formulation is not restricted, both in terms of the variables $X$ and $Y$, as well as in the compression variable $T$. It can be naturally extended to nominal and continuous variables, as well as dimension reduction techniques rather than clustering. This is the goal of the current paper.

The general treatment of IB for continuous $T$ yields the same set of self-consistent equations obtained already in [1]. But rather than solving them for the distributions $p(t|x)$, $p(t)$ and $p(y|t)$ using the generalized Blahut-Arimoto algorithm as proposed there, one can turn them into two coupled eigenvector problems for the logarithmic functional derivatives $\frac{\delta \log p(x|t)}{\delta t}$ and $\frac{\delta \log p(y|t)}{\delta t}$, respectively. Solving these equations, in general, turns out to be a rather difficult challenge. As in many other cases, however, the problem turns out to be analytically tractable when $X$ and $Y$ are joint multivariate Gaussian variables, as shown in this paper.

The optimal compression in the Gaussian Information Bottleneck (GIB) is defined in terms of the compression-relevance tradeoff, determined through the parameter $\beta$. It turns out to be a noisy linear projection to a subspace whose dimension is determined by the tradeoff parameter $\beta$. The subspaces are spanned by the basis vectors obtained in the well known *Canonical Correlation Analysis (CCA)*[3] method, but the exact nature of the projection is determined in a unique way via the tradeoff parameter $\beta$. Specifically, as $\beta$ increases, additional dimensions are added to the projection variable $T$, through a series of critical points (structural phase transitions), while at the same time the relative magnitude of each basis vector is rescaled. This process continues until all the relevant information about $Y$ is captured in $T$. This demonstrates how the IB formalism provides a continuous measure of model complexity in information theoretic terms.

The idea of maximization of relevant information was also taken in the *Imax* framework [4, 5]. In that setting, there are two feed forward networks with inputs $X_a$, $X_b$ and output neurons $Y_a$, $Y_b$. The output neuron $Y_a$ serves to define relevance to the output of the neighboring network $Y_b$. Formally, The goal is to tune the incoming weights of both output neurons, such that their mutual information $I(Y_a;Y_b)$ is maximized. An important difference between *Imax* and the IB setting, is that in the *Imax* setting, $I(Y_a;Y_b)$ is invariant to scaling and translation of the $Y$'s since the compression achieved in the mapping $X_a \rightarrow Y_a$ is not modeled explicitly. In contrast, the IB framework aims to characterize the dependence of the solution on the explicit compression term $I(T;X)$, which is a *scale sensitive* measure when the transformation is noisy. This view of compressed representation $T$ of the inputs $X$ is useful when dealing with neural systems that are stochastic in nature and limited in their response amplitudes and are thus constrained to finite $I(T;X)$.

## 2 Gaussian Information Bottleneck

We now formalize the problem of Information Bottleneck for Gaussian variables. Let $(X,Y)$ be two jointly Gaussian variables of dimensions $n_x, n_y$ and denote by

$\Sigma_x, \Sigma_y$ the covariance matrices of $X, Y$ and by $\Sigma_{xy}$ their cross-covariance matrix[1].

The goal of GIB is to compress the variable $X$ via a stochastic transformation into another variable $T \in R^{n_x}$, while preserving information about $Y$. With Gaussian $X$ and $Y$, the optimal $T$ is also jointly Gaussian with $X$ and $Y$. The intuition is that only second order correlations exist in the joint distribution $p(X, Y)$, so that distributions of $T$ with higher order moments do not carry additional information. This can be rigorously shown using an application of the entropy power inequality as in [6], and will be published elsewhere. Note that we do not explicitly limit the dimension of $T$, since we will show that the effective dimension is determined by the value of $\beta$. Since every two random variables $X, T$ with jointly Gaussian distribution can be presented as $T = AX + \xi$, where $\xi \sim N(0, \Sigma_\xi)$ is another Gaussian that is independent of $X$, we formalize the problem as the minimization

$$\min_{A, \Sigma_\xi} \mathcal{L} \equiv I(X; T) - \beta I(T; Y) \tag{2}$$

over the noisy linear transformations parametrized by the transformation $A$ and noise covariance $\Sigma_\xi$. $T$ is normally distributed $T \sim N(0, \Sigma_t)$ with $\Sigma_t = A\Sigma_x A^T + \Sigma_\xi$.

## 3   The optimal projection

A main result of this paper is the characterization of the optimal $A, \Sigma_\xi$ as a function of $\beta$

**Theorem 3.1**  *The optimal projection $T = AX + \xi$ for a given tradeoff parameter $\beta$ is given by $\Sigma_\xi = I_x$ and*

$$A = \begin{cases} [\mathbf{0}^T; \ldots; \mathbf{0}^T] & 0 \leq \beta \leq \beta_1^c \\ [\alpha_1 \mathbf{v}_1^T, \mathbf{0}^T; \ldots; \mathbf{0}^T] & \beta_1^c \leq \beta \leq \beta_2^c \\ [\alpha_1 \mathbf{v}_1^T; \alpha_2 \mathbf{v}_2^T; \mathbf{0}^T; \ldots, \mathbf{0}^T] & \beta_2^c \leq \beta \leq \beta_3^c \\ \vdots \end{cases} \tag{3}$$

*where $\{\mathbf{v}_1^T, \mathbf{v}_2^T, \ldots, \mathbf{v}_{n_x}^T\}$ are left eigenvectors of $\Sigma_{x|y}\Sigma_x^{-1}$ sorted by their corresponding ascending eigenvalues $\lambda_1, \lambda_2, \ldots, \lambda_{n_x}$, $\beta_i^c = \frac{1}{1-\lambda_i}$ are critical $\beta$ values, $\alpha_i$ are coefficients defined by $\alpha_i \equiv \frac{\beta(1-\lambda_i)-1}{\lambda_i r_i}$, $r_i \equiv \mathbf{v}_i^T \Sigma_x \mathbf{v}_i$, $\mathbf{0}^T$ is an $n_x$ dimensional row vector of zeros, and semicolons separate rows in the matrix $A$.*

This theorem asserts that the optimal projection consists of eigenvectors of $\Sigma_{x|y}\Sigma_x^{-1}$, combined in an interesting manner: For $\beta$ values that are smaller than the smallest critical point $\beta_1^c$, compression is more important than any information preservation and the optimal solution is the degenerated one $A \equiv 0$. As $\beta$ is increased, it goes through a series of critical points $\beta_i^c$, at each of which another eigenvector of $\Sigma_{x|y}\Sigma_x^{-1}$ is added to $A$. Even though the rank of $A$ increases at each of these transition points, it changes smoothly as a function of $\beta$ since at the critical point $\beta_i^c$ the coefficient $\alpha_i$ vanishes. Thus $\beta$ parameterizes a "continuous rank" of the projection.

To illustrate the form of the solution, we plot the landscape of the target function $\mathcal{L}$ together with the solution in a simple problem where $X \in R^2$ and $Y \in R$. In this case $A$ has a single non-zero row, thus $A$ can be thought of as a row vector

**Figure 1.** $\mathcal{L}$ as a function of all possible projections $A$, for $A : R^2 \to R$, obtained numerically from Eq. 4. Dark-red: low $\mathcal{L}$ values; light-yellow: large $\mathcal{L}$ values. $\Sigma_{xy} = [0.1\ 0.2]$, $\Sigma_x = I_2$. **A.** For $\beta = 15$, the optimal solution is the degenerated solution $A \equiv 0$. **B.** For $\beta = 100$, the eigenvector of $\Sigma_{x|y}\Sigma_x^{-1}$ with a norm according to theorem 3.1 (superimposed) is optimal.

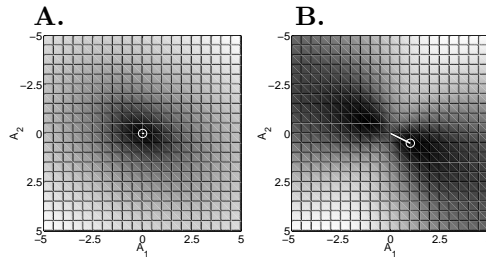

of length 2, that projects $X$ to a scalar $A : X \to R$, $T \in R$. Figure 1 shows the target function $\mathcal{L}$ as a function of the projection $A$. In this example, $\lambda_1 = 0.95$, thus $\beta_1^c = 20$. Therefor, for $\beta = 15$ (figure 1A) the zero solution is optimal, but for $\beta = 100 > \beta^c$ (figure 1B) the corresponding eigenvector is a feasible solution, and the target function manifold contains two mirror minima. As $\beta$ increases from 0 to $\infty$, these two minima, starting as a single unified minimum at zero, split at $\beta_1^c$, and then diverge apart to $\infty$.

We now turn to prove theorem 3.1[2]. We start by rewriting $\mathcal{L}$ using the formula for the entropy of a $d$ dimensional Gaussian variable $h(X) = \frac{1}{2}\log((2\pi e)^d|\Sigma_x|)$, where $|\cdot|$ denotes a determinant. Using the Schur complement formula to calculate the covariance of the conditional variable $T|Y$ we have $\Sigma_{t|y} = \Sigma_t - \Sigma_{ty}\Sigma_y^{-1}\Sigma_{yt} = A\Sigma_{x|y}A^T + \Sigma_\xi$, and the target function (up to a factor of 2) can be written as

$$\mathcal{L}(A, \Sigma_\xi) = (1-\beta)\log|A\Sigma_x A^T + \Sigma_\xi| - \log|\Sigma_\xi| + \beta\log|A\Sigma_{x|y}A^T + \Sigma_\xi| . \quad (4)$$

Although $\mathcal{L}$ is a function of both the noise $\Sigma_\xi$ and the projection $A$, it can be easily shown that for every pair $(A, \Sigma_\xi)$, there is another projection $\tilde{A} = \sqrt{D^{-1}}VA$ where $\Sigma_\xi = VDV^T$ and $\mathcal{L}(\tilde{A}, I) = \mathcal{L}(A, \Sigma_\xi)$ [3]. This allows us to simplify the calculations by replacing the noise covariance matrix $\Sigma_\xi$ with the identity matrix.

To identify the minimum of $\mathcal{L}$ we now differentiate $\mathcal{L}$ w.r.t. to the projection $A$ using the algebraic identity $\frac{\delta}{\delta A}\log|ACA^T| = (ACA^T)^{-1}2AC$ which holds for any symmetric matrix $C$. Equating this derivative to zero and rearranging, we obtain necessary conditions for an internal minimum of $\mathcal{L}$

$$(\beta - 1)/\beta\left[(A\Sigma_{x|y}A^T + I_d)(A\Sigma_x A^T + I_d)^{-1}\right]A = A\left[\Sigma_{x|y}\Sigma_x^{-1}\right] . \quad (5)$$

Equation 5 shows that the multiplication of $\Sigma_{x|y}\Sigma_x^{-1}$ by $A$ must reside in the span of the rows of $A$. This means that $A$ should be spanned by up to $d$ eigenvectors of $\Sigma_{x|y}\Sigma_x^{-1}$. We can therefore represent the projection $A$ as a mixture $A = WV$ where the rows of $V$ are left normalized eigenvectors of $\Sigma_{x|y}\Sigma_x^{-1}$ and $W$ is a mixing matrix that weights these eigenvectors. In the remainder of this section we characterize the nature of the mixing matrix $W$.

**Lemma 3.2** *The optimal mixing matrix $W$ is a diagonal matrix of the form*

$$W = diag\left[\sqrt{\frac{\beta(1 - \lambda_1) - 1}{\lambda_1 r_1}}\mathbf{v}_1^T, \ldots, \sqrt{\frac{\beta(1 - \lambda_k) - 1}{\lambda_k r_k}}\mathbf{v}_k^T, \mathbf{0}^T, \ldots, \mathbf{0}^T\right] \quad (6)$$

where $\{\mathbf{v}_1^T, \ldots, \mathbf{v}_k^T\}$ and $\{\lambda_1, \ldots, \lambda_k\}$ are $k \leq n_x$ eigenvectors and eigenvalues of $\Sigma_{x|y}\Sigma_x^{-1}$ with $\beta_1^c, \ldots, \beta_k^c \leq \beta$.

**Proof:** We write $V\Sigma_{x|y}\Sigma_x^{-1} = DV$ where $D$ is a diagonal matrix whose elements are the corresponding eigenvalues, and denote by $R$ the diagonal matrix whose $i^{th}$ element is $r_i = \mathbf{v}_i^T \Sigma_x \mathbf{v}_i$. When $k = n_x$, we substitute $A = WV$ into equation 5, and use the fact that $W$ is full rank to obtain

$$W^T W = [\beta(I - D) - I](DR)^{-1} . \tag{7}$$

While this does not uniquely characterize $W$, we note that if we substitute $A$ into the target function in equation 4, and use properties of the eigenvalues, we have

$$\mathcal{L} = (1 - \beta) \sum_{i=1}^{n} \log\left(||\mathbf{w}_i^T||^2 r_i + 1\right) + \beta \sum_{i=1}^{n} \log\left(||\mathbf{w}_i^T||^2 r_i \lambda_i + 1\right) \tag{8}$$

where $||\mathbf{w}_i^T||^2$ is the $i^{th}$ element of the diagonal of $W^T W$. This shows that $\mathcal{L}$ depends only on the norm of the columns of $W$, and all matrices $W$ that satisfy (7) yield the same target function. We can therefore choose to take $W$ to be the diagonal matrix which is the square root of (7)

$$W = \sqrt{[\beta(I - D) - I](DR)^{-1}} \tag{9}$$

To prove the case of $k < n_x$, consider a matrix $W$ that is a $k \times k$ matrix padded with zeros, thus it mixes only the first $k$ eigenvectors. In this case, calculation similar to that above gives the solution $A$ which has $n_x - k$ zero rows. To complete the proof, it remains to be shown that the above solution capture all extrema points. This point is detailed in [7] due to space considerations. $\square$

We have thus characterized the set of all minima of $\mathcal{L}$, and turn to identify which of them achieve the global minima.

**Corollary 3.3** *The global minimum of $\mathcal{L}$ is obtained with all $\lambda_i$ satisfying $\beta > \beta_i^c$*

**Proof:** Substituting the optimal $W$ of equation 9 into equation 8 yields $\mathcal{L} = \sum_{i=1}^{k}(\beta - 1)\log\lambda_i + \log(1 - \lambda_i) + f(\beta)$. Since $0 \leq \lambda \leq 1$ and $\beta \geq \frac{1}{1-\lambda}$, $\mathcal{L}$ is minimized by taking all the eigenvalues that satisfy $\beta > \frac{1}{(1-\lambda_i)}$. $\square$

Taken together, these observations prove that for a given value of $\beta$, the optimal projection is obtained by taking all the eigenvectors whose eigenvalues $\lambda_i$ satisfy $\beta \geq \frac{1}{1-\lambda_i}$, and setting their norm according to $A = WV$. This completes the proof of theorem 3.1.

# 4 The GIB Information Curve

The information bottleneck is targeted at characterizing the tradeoff between information preservation (accuracy of relevant predictions) and compression. Interestingly, much of the structure of the problem is reflected in the *information curve*, namely the *maximal* value of relevant preserved information (accuracy), $I(T;Y)$, as a function of the complexity of the representation of $X$, measured by $I(T;X)$. This curve is related to the rate-distortion function in lossy source coding, as well as to the achievability limit in channel coding with side-information [8]. It is shown to be concave in general [9], but its precise functional form depends on the joint

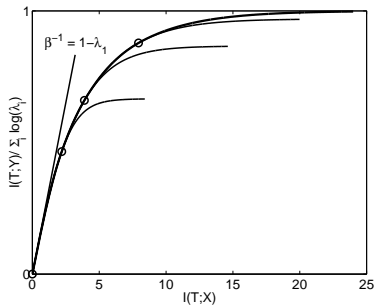

Figure 2. GIB information curve obtained with four eigenvalues $\lambda_i = 0.1, 0.5, 0.7, 0.9$. The information at the critical points are designated by circles. For comparison, information curves calculated with smaller number of eigenvectors are also depicted (all curves calculated for $\beta < 1000$). The slope of the curve at each point is the corresponding $\beta^{-1}$. The tangent at zero, with slope $\beta^{-1} = 1 - \lambda_1$, is super imposed on the information curve.

distribution and can reveal properties of the hidden structure of the variables. Analytic forms for the information curve are known only for very special cases, such as Bernoulli variables and some intriguing self-similar distributions. The analytic characterization of the Gaussian IB problem allows us to obtain a closed form expression for the information curve in terms of the relevant eigenvalues.

To this end, we substitute the optimal projection $A(\beta)$ into $I(T;X)$ and $I(T;Y)$ and isolate $I_\beta(T;Y)$ as a function of $I_\beta(T;X)$

$$
I_\beta(T;Y) = I_\beta(T;X) - \frac{n_I}{2} \log \left( \prod_{i=1}^{n_I} (1-\lambda_i)^{\frac{1}{n_I}} + e^{\frac{2I_\beta(T;X)}{n_I}} \prod_{i=1}^{n_I} \lambda_i^{\frac{1}{n_I}} \right) \tag{10}
$$

where the products are over the first $n_I$ eigenvalues, since these obey the critical $\beta$ condition, with $c_{n_I} \leq I_\beta(T;X) \leq c_{n_I+1}$ and $c_{n_I} = \frac{1}{2} \sum_{i=1}^{n_I-1} \log \frac{\lambda_{n_I}}{\lambda_i} \frac{1-\lambda_i}{1-\lambda_{n_I}}$.

The GIB curve, illustrated in Figure 2, is continuous and smooth, but is built of several of segments, since as $I(T;X)$ increases additional eigenvectors are used in the projection. The derivative of the curve is given by $\beta^{-1}$, which can be easily shown to be continuous and decreasing, yielding that the GIB information curve is concave everywhere. At each value of $I(T;X)$ the curve is therefore bounded by a tangent with a slope $\beta^{-1}(I(T;X))$. Generally in IB, the data processing inequality yields an upper bound on the slope at the origin, $\beta^{-1}(0) < 1$, in GIB we obtain a tighter bound: $\beta^{-1}(0) < 1 - \lambda_1$. The asymptotic slope of the curve is always zero, as $\beta \to \infty$, reflecting the law of diminishing return: adding more bits to the description of $X$ does not provide more accuracy about $T$. This interesting relation between the spectral properties of the covariance matrices raises interesting questions for special cases where more can be said about this spectrum, such as for patterns in neural-network learning problems.

## 5 Relation To Other Works

### 5.1 Canonical Correlation Analysis and Imax

The GIB projection derived above uses weighted eigenvectors of the matrix $\Sigma_{x|y}\Sigma_x^{-1} = I_x - \Sigma_{xy}\Sigma_y^{-1}\Sigma_{yx}\Sigma_x^{-1}$. The same eigenvectors are also used in *Canonical correlations Analysis* (CCA) [3], a statistical method that finds linear relations between two variables. CCA aims to find sets of basis vectors for the two variables that maximize the correlation coefficient between the projections of the variables on the basis vectors. The CCA bases are the eigenvectors of the matrices $\Sigma_y^{-1}\Sigma_{yx}\Sigma_x^{-1}\Sigma_{xy}$ and $\Sigma_x^{-1}\Sigma_{xy}\Sigma_y^{-1}\Sigma_{yx}$, and the square roots of their corresponding eigenvalues are termed *canonical correlation coefficients*. CCA was also shown to be a special case of continuous Imax [4, 5].

Although GIB and CCA involve the spectral analysis of the same matrices, they have some inherent differences. First of all, GIB characterizes not only the eigenvectors but also their norm, in a way that that depends on the trade-off parameter $\beta$. Since CCA depends on the correlation coefficient between the compressed (projected) versions of $X$ and $Y$, which is a normalized measure of correlation, it is invariant to a rescaling of the projection vectors. In contrast, for any value of $\beta$, GIB will choose one particular rescaling given by equation (4).

While CCA is symmetric (in the sense that both $X$ and $Y$ are projected), IB is non symmetric and only the $X$ variable is compressed. It is therefore interesting that both GIB and CCA use the same eigenvectors for the projection of $X$.

## 5.2   Multiterminal information theory

The Information Bottleneck formalism was recently shown [9] to be closely related to the problem of source coding with side information [8]. In the latter, two *discrete* variables $X, Y$ are encoded separately at rates $R_x, R_y$, and the aim is to use them to perfectly reconstruct $Y$. The bounds on the achievable rates in this case were found in [8] and can be obtained from the IB information curve.

When considering continuous variables, lossless compression at finite rates is no longer possible. Thus, mutual information for continuous variables is no longer interpretable in terms of encoding bits, but rather serves as an optimal measure of information between variables. The IB formalism, although coinciding with coding theorems in the discrete case, is more general in the sense that it reflects the tradeoff between compression and information preservation, and is not concerned with exact reconstruction. Such reconstruction can be considered by introducing distortion measures as in [6] but is not relevant for the question of finding representations which capture the information between the variables.

## 6   Discussion

We applied the information bottleneck method to continuous jointly Gaussian variables $X$ and $Y$, with a continuous representation of the compressed variable $T$. We derived an analytic optimal solution as well as a general algorithm for this problem (GIB) which is based solely on the spectral properties of the covariance matrices in the problem. The solution for GIB, characterized in terms of the trade-off parameter $\beta$, between compression and preserved relevant information, consists of eigenvectors of the matrix $\Sigma_{x|y}\Sigma_x^{-1}$, continuously adding up as weaker compression and more complex models are allowed. We provide an analytic characterization of the information curve, which relates the spectrum to relevant information in an intriguing manner. Besides its clean analytic structure, GIB offers a new way for analyzing empirical multivariate data when only its correlation matrices can be estimated. In thus extends and provides new information theoretic insight to the classical Canonical Correlation Analysis method.

The IB optima are known to obey three self consistent equations, that can be used in an iterative algorithm guaranteed to converge to a local optimum [1]. In GIB, these iterations over the conditional distributions $p(t|x)$, $p(t)$ and $p(y|t)$ can be transformed into iterations over the projection parameter $A$. In this case, the iterative IB algorithm turns into repeated projections on the matrix $\Sigma_{x|y}\Sigma_x^{-1}$, as used in power methods for eigenvector calculation. The parameter $\beta$ determines the scaling of the vectors, such that some of the eigenvectors decay to zero, while the others converge to their value defined in Theorem 3.1.

When handling real world data, the relevance variable $Y$ often contains multiple structures that are correlated to $X$, although many of them are actually irrelevant. The information bottleneck with side information ($IBSI$) [10] alleviates this problem using side information in the form of an *irrelevance* variable $Y^-$ about which information is removed. $IBSI$ thus aims to minimize $\mathcal{L} = I(X;T) - \beta\left(I(T;Y^+) - \gamma I(T;Y^-)\right)$. This functional can be analyzed in the case of Gaussian variables (*GIBSI: Gaussian IB with side information*), in a similar way to the analysis of GIB presented above. This results in a generalized eigenvalue problem involving the covariance matrices $\Sigma_{x|y^+}$ and $\Sigma_{x|y^-}$. The detailed solution of this problem as a function of the tradeoff parameters remains to be investigated.

For categorical variables, the IB framework can be shown to be closely related to maximum-likelihood in a latent variable model [11]. It would be interesting to see whether the GIB-CCA equivalence can be extended and give a more general understanding of the relation between IB and statistical latent variable models.

The extension of IB to continuous variables reveals a common principle behind regularized unsupervised learning methods ranging from clustering to CCA. It remains an interesting challenge to obtain practical algorithms in the IB framework for dimension reduction (continuous $T$) without the Gaussian assumption, for example by kernelizing [12] or adding non linearities to the projections (as in [13]).

## Footnotes

[1]For simplicity we assume that $\Sigma_x, \Sigma_y$ are full rank, otherwise $X, Y$ can be reduced to the proper dimensionality.

[2]Further details of the proofs can be found in a technical report [7].

[3]Although this theorem holds only for full rank $\Sigma_\xi$, it does not limit the generality of the discussion since low rank matrices yield infinite values of $\mathcal{L}$ and are therefore suboptimal.

## References

[1] N. Tishby, F.C. Pereira, and W. Bialek. The information bottleneck method. In *Proc. of 37th Allerton Conference on communication and computation*, 1999.

[2] N. Slonim. *Information Bottlneck theory and applications*. PhD thesis, Hebrew University of Jerusalem, 2003.

[3] H. Hotelling. The most predictable criterion. *Journal of Educational Psychology,*, 26:139–142, 1935.

[4] S. Becker and G.E. Hinton. A self-organizing neural network that discovers surfaces in random-dot stereograms. *Nature*, 355(6356):161–163, 1992.

[5] S. Becker. Mutual information maximization: Models of cortical self-organization. *Network: Computation in Neural Systems*, pages 7–31, 1996.

[6] T. Berger abd R. Zamir. A semi-continuous version of the berger-yeung problem. *IEEE Transactions on Information Theory*, pages 1520–1526, 1999.

[7] G. Chechik and A. Globerson. Information bottleneck and linear projections of gaussian processes. Technical Report 4, Hebrew University, May 2003.

[8] A.D. Wyner. On source coding with side information at the decoder. *IEEE Trans. on Info Theory*, IT-21:294–300, 1975.

[9] R. Gilad-Bachrach, A. Navot, and N. Tishby. An information theoretic tradeoff between complexity and accuracy. In *Proceedings of the COLT, Washington.*, 2003.

[10] G. Chechik and N. Tishby. Extracting relevant structures with side information. In S. Becker, S. Thrun, and K. Obermayer, editors, *Advances in Neural Information Processing Systems 15*, 2002.

[11] N. Slonim and Y. Weiss. Maximum likelihood and the information bottleneck. In S. Becker, S. Thrun, and K. Obermayer, editors, *Advances in Neural Information Processing Systems 15*, 2002.

[12] S. Mika, G. Ratsch, J. Weston, B. Scholkopf, A. Smola, and K. Muller. Invariant feature extraction and classification in kernel spaces. In S.A. Solla, T.K. Leen, and K.R. Muller, editors, *Advances in Neural Information Processing Systems 12*, 2000.

[13] A.J. Bell and T.J. Sejnowski. An information maximization approach to blind seperation and blind deconvolution. *Neural Computation*, 7:1129–1159, 1995.
